# Temporal Difference Based Actor Critic Learning - Convergence and Neural Implementation

**Dotan Di Castro, Dmitry Volkinshtein and Ron Meir**
Department of Electrical Engineering
Technion, Haifa 32000, Israel
{dot@tx},{dmitryv@tx},{rmeir@ee}.technion.ac.il

## Abstract

Actor-critic algorithms for reinforcement learning are achieving renewed popularity due to their good convergence properties in situations where other approaches often fail (e.g., when function approximation is involved). Interestingly, there is growing evidence that actor-critic approaches based on phasic dopamine signals play a key role in biological learning through cortical and basal ganglia loops. We derive a temporal difference based actor critic learning algorithm, for which convergence can be proved without assuming widely separated time scales for the actor and the critic. The approach is demonstrated by applying it to networks of spiking neurons. The established relation between phasic dopamine and the temporal difference signal lends support to the biological relevance of such algorithms.

## 1 Introduction

Actor-critic (AC) algorithms [22] were probably among the first algorithmic approaches to reinforcement learning (RL). In recent years much work focused on state, or state-action, value functions as a basis for learning. These methods, while possessing desirable convergence attributes in the context of table lookup representation, led to convergence problems when function approximation was involved. A more recent line of research is based on directly (and usually parametrically) representing the policy, and performing stochastic gradient ascent on the expected reward, estimated through trying out various actions and sampling trajectories [3, 15, 23]. However, such direct policy methods often lead to very slow convergence due to large estimation variance. One approach suggested in recent years to remedy this problem is the utilization of AC approaches, where the value function is estimated by a critic, and passed to an actor which selects an appropriate action, based on the approximated value function. The first convergence result for a policy gradient AC algorithm based on function approximation was established in [13], and extended recently in [5, 6]. At this stage it seems that AC based algorithms provide a solid foundation for provably effective approaches to RL based on function approximation. Whether these methods will yield useful solutions to practical problems remains to be seen.

RL has also been playing an increasingly important role in neuroscience, and experimentalists have directly recorded the activities of neurons while animals perform learning tasks [20], and used imaging techniques to characterize human brain activities [17, 24] during learning. It was suggested long ago that the basal ganglia, a set of ancient sub-cortical brain nuclei, are implicated in RL. Moreover, these nuclei are naturally divided into two components, based on the separation of the striatum (the main input channel to the basal ganglia) into the ventral and dorsal components. Several imaging studies [17, 24] have suggested that the ventral stream is associated with value estimation by a so called critic, while the dorsal stream has been implicated in motor output, action selection, and learning by a so called actor. Two further experimental findings support the view taken in this work.

First, it has been observed [20] that the short latency phasic response of the dopamine neurons in the midbrain strongly resembles the temporal difference (TD) signal introduced in theory of TD-learning [22], which can be used by AC algorithms for both the actor and the critic. Since mid-brain dopaminergic neurons project diffusively to both the ventral and dorsal components of the striatum, these results are consistent with a TD-based AC learning interpretation of the basal ganglia. Second, recent results suggest that synaptic plasticity occurring at the cortico-striatal synapses is strongly modulated by dopamine [18]. Based on these observations it has been suggested that the basal ganglia take part in TD based RL, with the (global) phasic dopamine signal serving as the TD signal [16] modulating synaptic plasticity.

Some recent work has been devoted to implementing RL in networks of spiking neurons (e.g., [1, 9, 12]). Such an approach may lead to specific and experimentally verifiable hypotheses regarding the interaction of known synaptic plasticity rules and RL. In fact, one tantalizing possibility is to test these derived rules in the context of ex-vivo cultured neural networks (e.g., [19]), which are connected to the environment through input (sensory) and output (motor) channels. We then envision dopamine serving as a biological substrate for implementing the TD signal in such a system. The work cited above is mostly based on direct policy gradient algorithms, (e.g., [3]), leading to non-AC approaches. Moreover, these algorithms were based directly on the reward, rather than on the biologically better motivated TD signal, which provides more information than the reward itself, and is expected to lead to improved convergence.

## 2   A Temporal Difference Based Actor-Critic Algorithm

The TD-based AC algorithm developed in this section is related to the one presented in [5, 6]. While the derivation of the present algorithm differs from the latter work (which also stressed the issue of the natural gradient) , the essential novel theoretical feature here is the establishment of convergence[1] without the restriction to two time scales which was used in [5, 6, 13]. This result is also important in a biological context, where, as far as we are aware, there is no evidence for such a time scale separation.

### 2.1   Problem Formulation

We consider a finite *Markov Decision Process* (MDP) in discrete time with a finite state set $\mathcal{X}$ of size $|\mathcal{X}|$ and a finite action set $\mathcal{U}$. The MDP models the environment in which the agent acts. Each selected action $u \in \mathcal{U}$ determines a stochastic matrix $P(u) = [P(y|x, u)]_{x,y \in \mathcal{X}}$ where $P(y|x, u)$ is the transition probability from a state $x \in \mathcal{X}$ to a state $y \in \mathcal{X}$ given the control $u$. A parameterized policy is described by a conditional probability function, denoted by $\mu(u|x, \theta)$, which maps observation $x \in \mathcal{X}$ into a control $u \in \mathcal{U}$ given a parameter $\theta \in \mathbb{R}^K$. For each state $x \in \mathcal{X}$ the agent receives a corresponding reward $r(x)$. The agent's goal is to adjust the parameter $\theta$ in order to attain maximum average reward over time.

For each $\theta \in \mathbb{R}^K$, we have a Markov Chain (MC) induced by $P(y|x, u)$ and $\mu(u|x, \theta)$. The state transitions of the MC are obtained by first generating an action $u$ according to $\mu(u|x, \theta)$, and then generating the next state according to $P(y|x, u)]_{x,y \in \mathcal{X}}$. Thus, the MC has a transition matrix $P(\theta) = [P(y|x, \theta)]_{x,y \in \mathcal{X}}$ which is given by $P(y|x, \theta) = \int_{\mathcal{U}} P(y|x, u) d\mu(u|x, \theta)$. We denote the set of these transition probabilities by $\mathcal{P} = \{P(\theta) | \theta \in \mathbb{R}^K\}$, and its closure by $\bar{\mathcal{P}}$. We denote by $P(x, u, y)$ the stationary probability to be in state $x$, choose action $u$ and go to state $y$. Several technical assumptions are required in the proofs below.

**Assumption 2.1.** *(i) Each MC $P(\theta)$, $P(\theta) \in \bar{\mathcal{P}}$, is aperiodic, recurrent, and contains a single equivalence class. (ii) The function $\mu(u|x, \theta)$ is twice differentiable. Moreover, there exist positive constants $B_r$ and $B_\mu$, such that for all $x \in \mathcal{X}$, $u \in \mathcal{U}$, $\theta \in \mathbb{R}^K$ and $1 \le k_1, k_2 \le K$, we have $|r(x)| \le B_r$, $|\partial\mu(u|x, \theta)/\partial\theta_k| \le B_\mu$, $|\partial^2\mu(u|x, \theta)/\partial\theta_{k_1}\theta_{k_2}| \le B_\mu$.*

As a result of assumption 2.1(i), we have the following lemma regarding the stationary distribution (Theorem 3.1 in [8]).

**Lemma 2.1.** *Under Assumption 2.1(i), each MC, $P(\theta) \in \bar{\mathcal{P}}$, has a unique stationary distribution, denoted by $\pi(\theta)$, satisfying $\pi(\theta)'P(\theta) = \pi(\theta)'$, where $x'$ is the transpose of vector $x$.*

Next, we define a measure for performance of an agent in an environment. The *average reward per stage* of a MC starting from an initial state $x_0 \in \mathcal{X}$ is defined by

$$J(x|\theta) \triangleq \lim_{T \to \infty} \mathrm{E}_\theta \left[ \frac{1}{T} \sum_{n=0}^{T} r(x_n) \Big| x_0 = x \right],$$

where $\mathrm{E}_\theta[\cdot]$ denotes the expectation under the probability measure $P(\theta)$, and $x_n$ is the state at time $n$. The agent's goal is to find $\theta \in \mathbb{R}^K$ which maximizes $J(x|\theta)$. The following lemma shows that under Assumption 2.1, the average reward per stage does not depend on the initial states (see Theorem 4.7 in [10]).

**Lemma 2.2.** *Under Assumption 2.1 and Lemma 2.1, the average reward per stage, $J(x|\theta)$, is independent of the starting state, is denoted by $\eta(\theta)$, and satisfies $\eta(\theta) = \pi(\theta)'r$.*

Based on Lemma 2.2, the agent's goal is to find a parameter vector $\theta$, which maximizes the average reward per stage $\eta(\theta)$. Performing the maximization directly on $\eta(\theta)$ is hard. In the sequel we show how this maximization can be performed by optimizing $\eta(\theta)$, using $\nabla \eta(\theta)$. A consequence of Assumption 2.1 and the definition of $\eta(\theta)$ is the following lemma (see Lemma 1 in [15]).

**Lemma 2.3.** *For each $x, y \in \mathcal{X}$ and for each $\theta \in \mathbb{R}^K$, the functions $P(y|x, \theta)$, $\pi(x|\theta)$, and $\eta(\theta)$, are bounded, twice differentiable, and have bounded first and second derivatives.*

Next, we define the *differential value function* of state $x \in \mathcal{X}$ which represents the average reward the agent receives upon starting from a state $x_0$ and reaching a recurrent state $x^*$ for the first time. Mathematically,

$$h(x|\theta) \triangleq \mathrm{E}_\theta \left[ \sum_{n=0}^{T} (r(x_n) - \eta(\theta)) \Big| x_0 = x \right], \tag{1}$$

where $T \triangleq \min\{k > 0 | x_k = x^*\}$. We define $h(\theta) \triangleq (h(x_1|\theta), \ldots, h(x_{|\mathcal{X}|}|\theta)) \in \mathbb{R}^{|\mathcal{X}|}$. For each $\theta \in \mathbb{R}^K$ and $x \in \mathcal{X}$, $h(x|\theta)$, $r(x)$, and $\eta(\theta)$ satisfy Poisson's equation (see Theorem 7.4.1 in [4]),

$$h(x|\theta) = r(x) - \eta(\theta) + \sum_{y \in \mathcal{X}} P(y|x, \theta) h(y|\theta). \tag{2}$$

Based on the differential value definition we define the *temporal difference* (TD) between the states $x \in \mathcal{X}$ and $y \in \mathcal{X}$. Formally,

$$d(x, y) \triangleq r(x) - \eta(\theta) + h(y|\theta) - h(x|\theta). \tag{3}$$

The TD measures the difference between the differential value estimate following the receipt of reward $r(x)$ and a move to a new state $y$, and the estimate of the current differential state value at state $x$.

## 2.2 Algorithmic details and single time scale convergence

We start with a definition of the *likelihood ratio derivative*, $\psi(x, u|\theta) \triangleq \nabla \mu(u|x, \theta)/\mu(u|x, \theta)$, which we assume to be bounded.

**Assumption 2.2.** *For all $x \in \mathcal{X}$, $u \in \mathcal{U}$, and $\theta \in \mathbb{R}^K$, there exists a positive constant, $B_\psi$, such that $|\psi(x, u|\theta)| \leq B_\psi < \infty$.*

In order to improve the agent's performance, we need to follow the gradient direction. The following theorem shows how the gradient of the average reward per stage can be calculated by the TD signal. Similar variants of the theorem were proved using the $Q$-value [23] or state value [15] instead of the TD-signal.

**Theorem 2.4.** *The gradient of the average reward per stage for $\theta \in \mathbb{R}^K$ can be expressed by*

$$\nabla \eta(\theta) = \sum_{x, y \in \mathcal{X}, u \in \mathcal{U}} P(x, u, y) \psi(x, u|\theta) \left(d(x, y) + f(x)\right) \qquad (f(x) \text{ arbitrary}). \tag{4}$$

The theorem was proved using an advantage function argument in [6]. We provide a direct proof in section A of the supplementary material. The flexibility resulting from the function $f(x)$ allows us to encode the TD signal using biologically realistic positive values only, without influencing the convergence proof. In this paper, for simplicity, we use $f(x) = 0$.

Based on Theorem 2.4, we suggest an TD-based AC algorithm. This algorithm is motivated by [15] where an actor only algorithm was proposed. In [15] the differential value function was re-estimated afresh for each regenerative cycle leading to a large estimation variance. Using the continuity of the actor's policy function in $\theta$, the difference between the estimates between regenerative cycles is small. Thus, the critic has a good initial estimate at the beginning of each cycle, which is used here in order to reduce the variance. A related AC algorithm was proposed in [5, 6], where two time scales were assumed in order to use Borkar's two time scales convergence theorem [7]. In our proposed algorithm, and associated convergence theorem, we do not assume different time scales for the actor and for the critic.

We present batch mode update equations[2] in Algorithm 1 for the actor and the critic. The algorithm is based on some recurrent state $x^*$; the visit times to this state are denoted by $t_0, t_1, \ldots$. Updated occur only at these times (batch mode). We define a cycle of the algorithm by the time indices which satisfy $t_m \leq n < t_{m+1}$. The variables $\tilde{d}$, $\tilde{h}(x)$, and $\tilde{\eta}$ are the critic's estimates for $d$, $h(x|\theta)$, and $\eta(\theta)$ respectively.

---

**Algorithm 1** Temporal Difference Based Actor Critic Algorithm

---

1: Given
   - An MDP with finite set $\mathcal{X}$ of states and a recurrent state $x^*$, satisfying 2.1(i).
   - Hitting times $t_0 < t_1 < t_2 < \cdots$ for the state $x^*$.
   - Step coefficients $\gamma_m$ such that $\sum_{m=1}^{\infty} \gamma_m = \infty$ and $\sum_{m=1}^{\infty} \gamma_m^2 < \infty$.
   - A parameterized policy $\mu(u|x, \theta)$, $\theta \in \mathbb{R}^K$, which satisfies Assumption 2.1(ii).
   - A set $H$, constants $B_{\tilde{h}}$ and $B_\theta$, and an operator $\Pi_H$ according to Assumption B.1.
   - Step parameters $\Gamma_\eta$ and $\Gamma_h$ satisfying Theorem 2.6.
2: Initiate the critic's variables:
   - $\tilde{\eta}_0 = 0$ (the estimate of the average reward per stage)
   - $\tilde{h}_0(x) = 0, \quad \forall x \in \mathcal{X}$ (the estimate of the differential value function)
3: Initiate the actor: $\theta_0 = 0$ and choose $f(x)$ (see (4))
4: **for** each state $x_{t_{m+1}}$ visited **do**
5:    **Critic:** For all $x \in \mathcal{X}$, $N_m(x) \triangleq \min\{t_m < k < t_{m+1}|x_k = x\}$, $(\min(\emptyset) = \infty)$

$$\tilde{d}(x_n, x_{n+1}) = r(x_n) - \tilde{\eta}_m + \tilde{h}_m(x_{n+1}) - \tilde{h}_m(x_n),$$

$$\tilde{h}_{m+1}(x) = \tilde{h}_m(x) + \gamma_m \Gamma_h \left( \sum_{n=N_m(x)}^{t_{m+1}-1} \tilde{d}(x_n, x_{n+1}) \right), \quad \forall x \in \mathcal{X},$$

$$\tilde{\eta}_{m+1} = \tilde{\eta}_m + \gamma_m \Gamma_\eta \sum_{n=t_m}^{t_{m+1}-1} (r(x_n) - \tilde{\eta}_m).$$

6:    **Actor:** $\theta_{m+1} = \theta_m + \gamma_m \sum_{n=t_m}^{t_{m+1}-1} \psi(x_n, u_n|\theta_m)(\tilde{d}(x_n, x_{n+1}) + f(x_n))$
7:    Project each component of $\tilde{h}_{m+1}$ and $\theta_{m+1}$ onto $H$ (see Assumption B.1.).
8: **end for**

---

In order to prove the convergence of Algorithm 1, we establish two basic results. The first shows that the algorithm converges to the set of ordinary differential equations (5), and the second establishes conditions under which the differential equations converge locally.

**Theorem 2.5.** *Under Assumptions 2.1 and B.1, Algorithm 1 converges to the following set of ODE's*

$$
\begin{cases}
\dot{\theta} = T(\theta)\nabla\eta(\theta) + C(\theta)\left(\eta(\theta) - \tilde{\eta}\right) + \sum_{x \in \mathcal{X}} D^{(x)}(\theta)\left(h(x|\theta) - \tilde{h}(x)\right), \\
\dot{\tilde{h}}(x) = \Gamma_h\left(h(x|\theta) - \tilde{h}(x)\right) + \Gamma_h T(\theta)\left(\eta(\theta) - \tilde{\eta}\right), \quad x \in \mathcal{X} \\
\dot{\tilde{\eta}} = \Gamma_\eta T(\theta)\left(\eta(\theta) - \tilde{\eta}\right),
\end{cases}
\tag{5}
$$

*with probability* 1*, where*

$$
T = \min\{k > 0 | x_0 = x^*, x_k = x^*\}, \quad T(\theta) = \mathrm{E}_\theta[T], \quad C(\theta) = \mathrm{E}_\theta\left[\sum_{n=0}^{T-1} \psi(x_n, u_n|\theta)\Big| x_0 = x^*\right],
$$

$$
D^{(x)}(\theta) = \mathrm{E}_\theta\left[\sum_{n=0}^{T-1} \mathbb{1}\{x_{n+1} = x\}\,\psi(x_n, u_n|\theta)\Big| x_0 = x^*\right] + \mathrm{E}_\theta\left[\sum_{n=0}^{T-1} (\mathbb{1}\{x_n = x\}\,\psi(x_n, u_n|\theta)\Big| x_0 = x^*\right],
$$

*and where* $T(\theta)$*,* $C(\theta)$*, and* $D^{(x)}(\theta)$ *are continuous with respect to* $\theta$*.*

Theorem 2.5 is proved in section B of the supplementary material, based on the theory of stochastic approximation, and more specifically, on Theorem 5.2.1 in [14]. An advantage of the proof technique is that it does not need to assume two time scales.

The second theorem, proved in section C of the supplementary material, states the conditions for which $\eta(\theta_t)$ converges to a ball around the local optimum.

**Theorem 2.6.** *If we choose* $\Gamma_\eta \geq B_{\tilde{\eta}}^2/\epsilon_\eta$ *and* $\Gamma_h \geq B_{\tilde{h}}^2/\epsilon_h$*, for some positive constants* $\epsilon_h$ *and* $\epsilon_\eta$*, then* $\limsup_{t\to\infty} \|\nabla\eta(\theta(t))\| \leq \epsilon$*, where* $\epsilon \triangleq B_C\epsilon_\eta + |\mathcal{X}|B_D\epsilon_h$*. The constants* $B_{\dot{\eta}}$ *and* $B_{\dot{h}}$ *are defined in Section C of the supplementary material.*

## 3 A Neural Algorithm for the Actor Using McCulloch-Pitts Neurons

In this section we apply the previously developed algorithm to the case of neural networks. We start with the classic binary valued McCulloch-Pitts neuron, and then consider a more realistic spiking neuron model. While the algorithm presented in Section 2 was derived and proved to converge in batch mode, we apply it here in an online fashion. The derivation of an online learning algorithm from the batch version is immediate (e.g., [15]), and a proof of convergence in this setting is currently underway.

**A McCulloch-Pitts actor network**

The dynamics of the binary valued neurons, given at time $n$ by $\{u_i(n)\}_{i=1}^N$, $u_i(n) \in \{0, 1\}$, is assumed to be based on stochastic discrete time parallel updates, given by

$$
\Pr(u_i(n) = 1) = \sigma(v_i(n)) \quad \text{where} \quad v_i(n) = \sum_{j=1}^N w_{ij}u_j(n-1) \qquad (i = 1, 2, \ldots, N).
$$

Here $\sigma(v) = 1/(1 + \exp(-v))$, and the parameters $\theta$ in Algorithm 1 are given by $\{w_{ij}\}$, where $w_{ij}(n)$ is the $j \mapsto i$ synaptic weight at time $n$. Each neuron's stochastic output $u_i$ is viewed as an action.

Applying the actor update from Algorithm 1 we obtain the following online learning rule

$$
w_{ij}(n+1) = w_{ij}(n) + \gamma d(x(n), x(n+1))\left(u_i(n) - \sigma(v_i(n))\right) u_j(n-1).
\tag{6}
$$

where $d(x(n), x(n+1))$ is the TD signal.

The update (6) can be interpreted as an error-driven Hebbian-like learning rule modulated by the TD signal. It resembles the direct policy update rule presented in [2], except that in this rule the reward signal is replaced by the TD signal (computed by the critic). Moreover, the eligibility trace formalism in [2] differs from our formulation.

We describe a simulation experiment conducted using a one layered feed-forward artificial neural network which functions as an actor, combined with a non biologically motivated critic. The purpose of the experiment is to examine a simple neuronal model, using different actor and critic architectures. The actor network consists of a single layered feed-forward network of McCulloch-Pitts neurons, and TD modulated synapses as described above, where the TD signal is calculated by a critic. The environment is a maze with barriers consisting of 36 states, see Figure 1(b), where a reward of value 1 is provided at the top right corner, and is zero elsewhere. Every time the agent receives a reward, it is transferred randomly to a different location in the maze. At each time step, the agent is given an input vector which represents the state. The output layer consists of 4 output neurons where each neuron represents an action from the action set $\mathcal{U} = \{\text{up}, \text{down}, \text{left}, \text{right}\}$. We used two different input representations for the actor, consisting either of 12 or 36 neurons (note that the minimum number of input neurons to represent 36 states is 6, and the maximum number is 36). The architecture with 36 input neurons represents each maze state with one exclusive neuron, thus, there is no overlap between input vectors. The architecture with 12 input neurons uses a representation where each state is represented by two neurons, leading to overlaps between the input vectors. We tested two types of critic: a table based critic which performs iterates according to Algorithm 1, and an exact TD which provides the TD of the optimal policy. The results are shown in Figure 1(c), averaged over 25 runs, and demonstrate the importance of good input representations and precise value estimates.

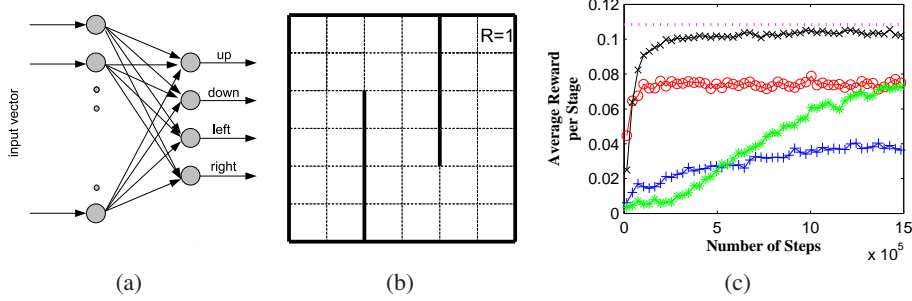

|  |  |  |
|:---:|:---:|:---:|
| (a) | (b) | (c) |

Figure 1: (a) A illustration of the McCulloch-Pitts network. (b) A diagram of the maze where the agent needs to reach the reward at the upper right corner. (c) The average reward per stage in four different cases: an actor consisting of 12 input neurons and a table based critic (blue crosses), an actor consisting of 36 input neurons and a table based critic (green stars), an actor consisting of 12 input neurons and exact critic (red circles), and an actor consisting of 36 input neurons and an exact TD (black crosses). The optimal average reward per stage is denoted by the dotted line, while a random agent achieves a reward of 0.005.

**A spiking neuron actor**

Actual neurons function in continuous time producing action potentials. In extension of [1, 9], we developed an update rule which is based on the Spike Response Model (SRM) [11]. For each neuron we define a state variable $v_i(t)$ which represents the *membrane potential*. The dynamics of $v_i(t)$ is given by

$$v_i(t) = \vartheta_i(t - \hat{t}_i) + \sum_{j=1}^{N} w_{ij}(t) \sum_{t_j^f} \epsilon_{ij}(t - \hat{t}_i, t - t_j^f), \tag{7}$$

where $w_{ij}(t)$ is the synaptic efficacy, $\hat{t}_i$ is the last spike time of neuron $i$ prior o $t$, $\vartheta_i(t)$ is the refractory response, $t_j^f$ are the times of the presynaptic spikes emitted prior to time $t$, and $\epsilon_{ij}(t - \hat{t}_i, t - t_j^f)$ is the response induced by neuron $j$ at neuron $i$. The second summation in (7) is over all spike times of neuron $j$ emitted prior to time $t$. The neuron model is assumed to have a noisy threshold, which we model by an *escape noise model* [11]. According to this model, the neuron fires in the time interval $[t, t + \delta t)$ with probability $u_i(t)\delta t = \rho_i(v_i(t) - v_{\text{th}})\delta t$, where $v_{\text{th}}$ is the firing threshold and $\rho_i(\cdot)$ is a monotonically increasing function. When the neuron reaches the threshold it is assumed to fire and the membrane potential is reset to $v_{\text{r}}$.

We consider a network of continuous time neurons and synapses. Based on Algorithm 1, using a small time step $\delta t$, we find

$$w_{ij}(t + \delta t) = w_{ij}(t) + \gamma d(t)\psi_{ij}(t). \tag{8}$$

We define the output of the neuron (interpreted as an action) at time $t$ by $u_i(t)$. We note that the neuron's output is discrete and that at each time $t$, a neuron can fire, $u_i(t) = 1$, or be quiescent, $u_i(t) = 0$. Using the definition of $\psi$ from Section 2.2, yields (similar to [9])

$$\psi_{ij}(t) = \begin{cases} \frac{\rho_i'(t)}{\rho_i(t)} \sum_{\mathcal{H}_j^t} \epsilon_{ij}(t - \hat{t}_i, t - t_j^f), & \text{if } u_i(t) = 1 \\ -\frac{\delta t \rho_i'(t)}{1 - \delta t \rho_i(t)} \sum_{\mathcal{H}_j^t} \epsilon_{ij}(t - \hat{t}_i, t - t_j^f), & \text{if } u_i(t) = 0 \end{cases}$$

Taking the limit $\delta t \to 0$, yields the following continuous time update rule

$$\frac{dw_{ij}(t)}{dt} = \gamma d(t) \overbrace{\left( (1/\rho_i(t)) \sum_{\mathcal{H}_i} \delta(t - t_i^f) - 1 \right) \rho_i'(t)}^{F_{\text{post}}(\{t_i^f\})} \overbrace{\sum_{\mathcal{H}_j^t} \epsilon_{ij}(t - \hat{t}_i, t - t_j^f)}^{F_{\text{pre}}(\{t_j^f\})}. \tag{9}$$

Similarly to [1, 9] we interpret the update rule (9) as a TD modulated spike time dependent plasticity rule. A detailed discussion and interpretation of this update in a more biological context will be left to the full paper.

We applied the update rule (9) to an actor network consisting of spiking neurons based on (7). The network's goal was to reach a circle at the center of a 2D plain =, where the agent can move, using Newtonian dynamics, in the four principle directions. The actor is composed of an input layer and a single layer of modifiable weights. The input layer consists of 'sensory' neurons which fire according to the agent's location in the environment. The synaptic dynamics of the actor is determined by (9). The critic receives the same inputs as the actor, but uses a linear function approximation architecture rather than the table lookup used in Algorithm 1. A standard parameter update rule appropriate for this architecture (e.g., ch. 8 in [22]) was used to update the critic's parameters[3]. The output layer of the actor consists of four neuronal groups, representing the directions in which the agent can move, coded based on a firing rate model using Gaussian tuning curves. The TD signal is calculated according to (3). Whenever it reaches the centered circle, it receives a reward, and is transferred randomly to a new position in the environment.

Results of such a simulation are presented in Figure 3. Figure 3-a displays the agent's typical random walk like behavior prior to learning, . Figure 3-b depicts four typical trajectories representing the agent's actions after a learning phase. Finally, Figure 3-c demonstrates the increase of the average reward per stage, $\eta$, vs. time.

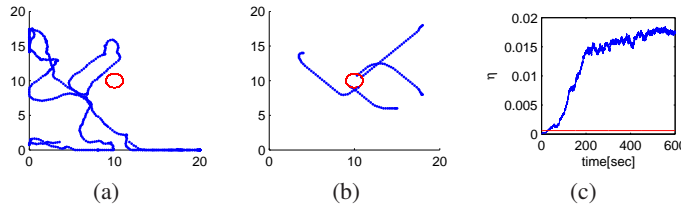

(a)        (b)        (c)

Figure 2: (a) Typical agent tracks prior to learning. (b) Agent trajectories following learning. (c) Average reward per stage plotted against time.

## 4 Discussion

We have presented a temporal difference based actor critic learning algorithm for reinforcement learning. The algorithm was derived from first principles based on following a noisy gradient of the

average reward, and a convergence proof was presented without relying on the widely used two time scale separation for the actor and the critic. The derived algorithm was applied to neural networks, demonstrating their effective operation in maze problems. The motivation for the proposed algorithm was biological, providing a coherent computational explanation for several recently observed phenomena: actor critic architectures in the basal ganglia, the relation of phasic dopaminergic neuromodulators to the TD signal, and the modulation of the spike time dependent plasticity rules by dopamine. While a great deal of further work needs to be done on both the theoretical and biological components of the framework, we hope that these results provide a tentative step in the (noisy!) direction of explaining biological RL.

## Footnotes

[1]Throughout this paper convergence refers to convergence to a small ball around a stationary point; see Theorem 2.6 for a precise definition.

[2]In order to prove convergence certain boundedness conditions need to be imposed, which appear as step 7 in the algorithm. For lack of space, the precise definition of the set $H$ is given in Assumption B.1 of the supplementary material.

[3]Algorithm 1 relies on a table lookup critic, while in this example we used a function approximation based critic, due to the large (continuous) state space.

## References

[1] D. Baras and R. Meir. Reinforcement learning, spike time dependent plasticity and the bcm rule. *Neural Comput.*, 19(8):22452279, 2007

[2] J. Baxter and P.L. Bartlett. Hebbian synaptic modifications in spiking neurons that learn. (Technical rep.). Canberra: Research School of Information Sciences and Engineering, Australian National University, 1999.

[3] J. Baxter and P.L. Bartlett. Infinite-Horizon Policy-Gradient Estimation. *J. of Artificial Intelligence Research*, 15:319–350, 2001.

[4] D.P. Bertsekas. *Dynamic Programming and Optimal Control, Vol I.*, 3rd Ed. Athena Scinetific, 2006.

[5] S. Bhatnagar, R. Sutton, M. Ghavamzadeh, and M. Lee. Incremental natural actor-critic algorithms. In J.C. Platt, D. Koller, Y. Singer, and S. Roweis, editors, *Advances in Neural Information Processing Systems 20*, pages 105–112. MIT Press, Cambridge, MA, 2008.

[6] S. Bhatnagar, R.S. Sutton, M. Ghavamzadeh, and M. Lee. Natural actor-critic algorithms. *Automatica*, To appear, 2008.

[7] V.S. Borkar. Stochastic approximation with two time scales. *Syst. Control Lett.*, 29(5):291294, 1997.

[8] P. Bremaud. *Markov Chains: Gibbs Fields, Monte Carlo Simulation, and Queues.* Springer, 1999.

[9] R.V. Florian. Reinforcement learning through modulation of spike-timing-dependent synaptic plasticity. *Neural Computation*, 19:14681502, 2007.

[10] R.G. Gallager. Discrete Stochastic Processes. Kluwer Academic Publishers, 1995.

[11] W. Gerstner and W.M. Kistler. *Spinking Neuron Models.* Cambridge University Press, Cambridge, 2002.

[12] E.M. Izhikevich. Solving the Distal Reward Problem through Linkage of STDP and Dopamine Signaling. *Cerebral Cortex*, 17(10):2443-52, 2007.

[13] V.R. Konda and J. Tsitsiklis. On actor critic algorithms. *SIAM J. Control Optim.*, 42(4):11431166, 2003.

[14] H.J. Kushner and G.G. Yin. *Stochastic Approximation Algorithms and Applications.* Springer, 1997.

[15] P. Marbach and J. Tsitsiklis. Simulation-Based Optimization of Markov Reward Processes. *IEEE. Trans. Auto. Cont.*, 46:191–209, 1998.

[16] P.R. Montague, P. Dayan, and T.J. Sejnowski. A framework for mesencephalic dopamine systems based on predictive hebbian learning. *Journal of Neuroscience*, 16:19361947, 1996.

[17] J. ODoherty, P. Dayan, J. Schultz, R. Deichmann, K. Friston, and R.J. Dolan. Dissociable roles of ventral and dorsal striatum in instrumental conditioning. *Science*, 304:452454, 2004.

[18] J.N.J. Reynolds and J.R. Wickens. Dopamine-dependent plasticity of corticostriatal synapses. *Neural Networks*, 15(4-6):507521, 2002.

[19] S. Marom and G. Shahaf. Development, learning and memory in large random networks of cortical neurons: lessons beyond anatomy. *Quarterly Reviews of Biophysics*, 35:6387, 2002.

[20] W. Schultz. Multiple reward signals in the brain. *Nature Reviews Neuroscience*, 1:199207, Dec. 2000.

[21] S. Singh and P. Dayan. Analytical mean squared error curves for temporal difference learning. *Machine Learning*, 32:540, 1998.

[22] R. S. Sutton and A. G. Barto. *Reinforcement Learning*. MIT Press, 1998.

[23] R. Sutton, D. McAllester, S. Singh and Y. Mansour. Policy-Gradient Methods for Reinforcement Learning with Function Approximation. *Advances in Neural Information Processing Systems*, 12:1057–1063, 2000.

[24] E.M. Tricomi, M.R. Delgado, and J.A. Fiez. Modulation of caudate activity by action contingency. *Neuron*, 41(2):281292, 2004.
